# Over-complete representations on recurrent neural networks can support persistent percepts

**Shaul Druckmann**
Janelia Farm Research Campus
Howard Hughes Medical Institute
Ashburn, VA 20147
druckmanns@janelia.hhmi.org

**Dmitri B. Chklovskii**
Janelia Farm Research Campus
Howard Hughes Medical Institute
Ashburn, VA 20147
mitya@janelia.hhmi.org

## Abstract

A striking aspect of cortical neural networks is the divergence of a relatively small number of input channels from the peripheral sensory apparatus into a large number of cortical neurons, an over-complete representation strategy. Cortical neurons are then connected by a sparse network of lateral synapses. Here we propose that such architecture may increase the persistence of the representation of an incoming stimulus, or a percept. We demonstrate that for a family of networks in which the receptive field of each neuron is re-expressed by its outgoing connections, a represented percept can remain constant despite changing activity. We term this choice of connectivity REceptive FIeld REcombination (REFIRE) networks. The sparse REFIRE network may serve as a high-dimensional integrator and a biologically plausible model of the local cortical circuit.

## 1   Introduction

Two salient features of cortical networks are the numerous recurrent lateral connections within a cortical area and the high ratio of cortical cells to sensory input channels. In their seminal study [1], Olshausen and Field argued that such architecture may subserve sparse over-complete representations, which maximize representation accuracy while minimizing the metabolic cost of spiking. In this framework, lateral connections between neurons with correlated receptive fields mediate explaining away of the sensory input features[2]. With the exception of an Ising-like generative model for the lateral connections [3] and a mutual information maximization approach [4], most theoretical work on lateral connections did not focus on the representation over-completeness [5] and references therein.

Here, we propose that over-complete representations on recurrently connected networks offer a solution to a long-standing puzzle in neuroscience, that of maintaining a stable sensory percept in the absence of time-invariant persistent activity (rate of action potential discharge). In order for sensory percepts to guide actions, their duration must extend to behavioral time scales, hundreds of milliseconds or seconds if not more. However, many cortical neurons exhibit time-varying activity even during working memory tasks [6, 7] and references therein. If each neuron codes for orthogonal directions in stimulus space, any change in the activity of neurons would cause a distortion in the network representation, implying that a percept cannot be maintained.

We point out that, in an over-complete representation, network activity can change without any change in the percept, allowing persistent percepts to be maintained in face of variable neuronal activity. This results from the fact that the activity space has a higher dimensionality than that of the stimulus space. When the activity changes in a direction nulled by the projection onto stimulus space, the percept remains invariant.

What lateral connectivity can support persistent percepts, even in the face of changing neuronal activity? We derive the condition on lateral connection weights for networks to maintain persistent percepts, thus defining a family of REceptive FIeld REcombination networks. Furthermore, we propose that minimizing synaptic volume cost favors sparse REFIRE networks, whose properties are remarkably similar to that of the cortex. Such REFIRE networks act as high dimensional integrators of sensory input.

## 2  Model

We consider $n$ sensory neurons, their activity marked by $s$ in $\mathcal{R}^n$ which project to a layer of $m$ cortical neurons, where $m > n$. The activity of the $m$ neurons, marked by $a$ in $\mathcal{R}^m$, at any given time represents a percept of a certain stimulus. The represented percept $s$ is a linear superposition of feature vectors, stacked as columns of matrix $D$, weighted by the neuronal activity $a$:

$$s = Da. \tag{1}$$

For instance, $s$ could represent the intensity level of pixels in a patch of the visual field and the columns of $D$ a dictionary chosen to represent the patches, e.g. a set of Gabor filters [8]. Since $m > n$, the columns of dictionary $D$ cannot be orthogonal and hence define a frame rather than a basis [9].

### 2.1  Frames

A frame is a generalization of the idea of a basis to linearly dependent elements [9]. The mapping between the activity space $\mathcal{R}^m$ and the sensory space $\mathcal{R}^n$ is accomplished by the synthesis operator, $D$. The adjoint operator $D^T$ is called the analysis operator and their composition the frame operator $DD^T$. As a consequence of columns of $D$ being a frame, a given vector in the space of percepts can be represented non-uniquely, i.e. with different coefficients expressed by neuronal activity $a$. The general form of coefficients is given by:

$$a = D^T(DD^T)^{-1}s + a_\perp, \tag{2}$$

where $a_\perp$ belongs to the null-space of $D$, i.e. $Da_\perp = 0$.

One choice of coefficients, called frame coefficients, corresponds to $a_\perp = 0$ and minimizes their $l_2$ norm. Alternatively one can choose a set of coefficients minimizing the $l_1$ norm. These can be computed by Matching Pursuit [10], Basis Pursuit [11] or LASSO [12], or by the dynamics of a neural network with feedforward and lateral connections [13]. In summary, the neural activity is an over-complete representation of the sensory percepts, the $m$ columns of $D$ acting as a frame for the space of sensory percepts.

### 2.2  Persistent percepts and lateral connectivity

Now, we derive a necessary and sufficient condition on the lateral connections $L$ such that for every $a$ the percept represented by Equation (1) persists. We focus on the dynamics of $a$ following a transient presentation of the sensory stimulus. The dynamics of a network with lateral connectivity matrix $L$ is given by:

$$\dot{a} = -a + La, \tag{3}$$

where time is measured in units of the neuronal membrane time constant. Requiring time-invariant persistent activity amounts to $\dot{a} = 0$ or

$$a = La. \tag{4}$$

However, this is not necessary if we require only the *percept* represented by the network to be fixed. Instead,

$$\dot{s} = D\dot{a} = D(-a + La) = 0 \tag{5}$$

Thus, setting the derivative of s to zero is tantamount to

$$Da = DLa. \tag{6}$$

If we require persistent percepts for any $a$, then:

$$D = DL \tag{7}$$

Equation (7) has a trivial solution $L = I$, which corresponds to a network with no actual lateral connections and only autapses. We do not consider this solution further for two reasons. First, autapses are extremely rare among cortical neurons[14]. Second, recurrent networks better support persistency than autapses [15, 16].

The intuition behind the derivation of Equation (7) is as follows: as the activity of each neuron changes due to the first term in the rhs of Equation (5) its contribution to the percept may change. To compensate for this change without necessarily keeping the activity fixed, we require that the other neurons adjust their activity according to Equation (6).

The condition imposed by Equation (7) on the synaptic weights can be understood as follows. For each neuron $j$ the sum of its post-synaptic partners receptive fields, weighted by the synaptic efficacy from neuron $j$ to the other neurons equals to the receptive field of neuron $j$. Thus, the other neurons get excited by exactly the amount that it would take for them to replace the lost contribution to the percept. Equation (7) and its non-trivial solutions that maintain persistent percepts are the main results of the present study. We term non-trivial solutions of Equation (7) REceptive FIeld RE-expression, or REFIRE networks due to the intuition underlying their definition.

Some patterns of activity satisfying Equation (4) will remain time-invariant themselves. These correspond to patterns spanned by the right eigenvectors of $L$ with an eigenvalue of one. Note that in order to satisfy Equation (7) a right eigenvector $v$ of L must have either an eigenvalue of one or be in the null-space of D.

There are infinitely many solutions satisfying Equation (7), since there are $m * n$ equations and $m * m$ variables in $L$. A general solution is given by:

$$L = D^T (DD^T)^{-1} D + L_\perp, \tag{8}$$

where $L_\perp$ indicates a component in $L$ corresponding to the null-space of $D$ i.e. $DL_\perp = 0$. We shall use these degrees of freedom to require a zero diagonal for $L$, thus avoiding autapses.

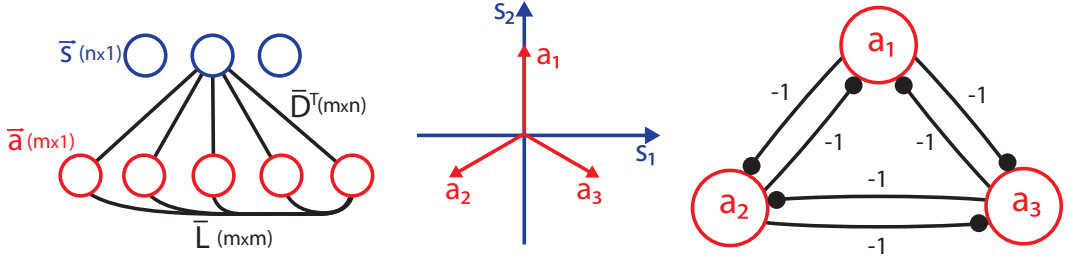

Figure 1: Schematic network diagram and Mercedes-Benz example. Left: Network diagram. Middle: Directions of vectors in the MB example. Right: visualization of $L$

## 2.3 An example: the Mercedes-Benz frame

In order to present a more intuitive view of the concept of persistent percepts we consider the Mercedes-Benz frame [17]. This simple frame spans the $\mathcal{R}^2$ plane with three frame elements: $[0 \quad 1], [-\sqrt{3}/2 \quad -1/2], [\sqrt{3}/2 \quad -1/2]$. In this case, the frame operator $DD^T$ has a particularly simple form, being proportional to the identity matrix, indicating that the frame is tight. The first term in the general form of $L$ (Equation (8)) has a non-zero diagonal, which can be removed by adding $L_\perp$, a matrix with all its entries equal to one (times a scalar). Thus, $L$ is:

$$L = \begin{pmatrix} 0 & -1 & -1 \\ -1 & 0 & -1 \\ -1 & -1 & 0 \end{pmatrix}$$

This seems a rather unlikely candidate matrix to support persistent percepts. However, consider starting out with the vector $a_0 = [1 \quad 0 \quad 0]$ representing the point $[0 \quad 1]$ on the plane, after convergence of the dynamics we have $a = [2/3 - 1/3 - 1/3]$. This new activity vector represents

exactly the same point on the plane: $Da = \begin{bmatrix} 0 & 1 \end{bmatrix}$. Thus, the percept, the point on the plane, remained constant despite changing neuronal activity. Note that some patterns of activity will remain strictly persistent themselves. These correspond to vectors which are a linear combination of the right eigenvectors of $L$ with an eigenvalue of one. In this case, these eigenvectors are: $v_1 = \begin{bmatrix} -1 & 1 & 0 \end{bmatrix}$, $v_2 = \begin{bmatrix} 1/2 & 1/2 & -1 \end{bmatrix}$.

### 2.4 The sparse REFIRE network

Which members of the family of REFIRE networks obeying equation (7) are most likely to model cortical networks? In the cortex, the connectivity is sparse and the synaptic weights are distributed exponentially [18, 19]. These measurements are consistent with minimizing cost proportional to synaptic weight, such as for example their volume. Motivated by these observations, we choose each column of $L$ as a sparse representation of each individual dictionary element by every *other* element. Define $\mathbf{D_j} = \mathbf{d_1}, \mathbf{d_2}, \ldots \mathbf{d_{j-1}}, \mathbf{d_{j+1}} \ldots \mathbf{d_m}$. We shall denote the sparse approximation coefficients by $\beta$. Therefore:

$$\beta_j^* = \min_{\beta_{\mathbf{j}} \in \mathcal{R}^{m-1}} ||\mathbf{d_j} - \mathbf{D_j}\beta_{\mathbf{j}}||_2^2 + \lambda ||\beta_{\mathbf{j}}||_1 \qquad (9)$$

These are vectors in $\mathcal{R}^{m-1}$, we now need to insert a zero in the position of the dictionary element that was extracted for each of these vectors. Denote by $\tilde{\beta}_{\mathbf{j}}$ a vector where a zero before the $j_{th}$ location of $\beta_{\mathbf{j}}$, resulting in a vector in $\mathcal{R}^m$. The connectivity of our model network is given by $\mathbf{L} = [\tilde{\beta}_{\mathbf{1}}, \tilde{\beta}_{\mathbf{2}}, \ldots \tilde{\beta}_{\mathbf{m}}]$ in $\mathcal{R}^{m \times m}$.

We call this form of $L$ the sparse REFIRE network. Similar networks were previously constructed on the raw data (or image patches) [20, 21], while sparse REFIRE networks reflect the relationship among dictionary elements. Previously, the dependencies between dictionary elements were captured by tree-graphs [22, 23].

## 3 Results

In this section, we apply our model to the primary visual cortex by modeling the receptive fields following the approach of [1]. We study the properties of the resulting sparse REFIRE network and compare them with experimentally established properties of cortical networks.

### 3.1 Constructing the sparse REFIRE network for visual cortex

We learn the sparse REFIRE network from a standard set of natural images [8]. We extract patches of size 13x13 pixels. We use a set of 100,000 such patches distributed evenly across different natural images to learn the model. Whitening was performed through PCA, after the DC component of each patch was removed. The dimensionality was reduced from 169 to 84 dimensions. We learn a four times over-complete dictionary, via the SPAMS online sparse approximation toolbox [24]. Figure 2 left shows the forward weights (columns of $D$) learned. As expected, the filters obtained are edge detectors differing in scale, spatial location and orientation.

The sparse REFIRE network was then learned from the dictionary using the same toolbox. Parameter $\lambda$ in equation (9) governs the tradeoff between sparsity and reconstruction fidelity, figure 2 right. We verified that the results presented in this study do not qualitatively change over a wide range of $\lambda$ and chose the value of $\lambda$ where the average probability of connection was 9%, in agreement with the experimental number of approximately 10%. For this choice the relative reconstruction mismatch was approximately $10^{-3}$. The distribution of synaptic weights in the network, Figure 3 left, shows a strong bias to zero valued connections and a heavier than gaussian tail as does the cortical data [25]. For an enlarged view of the network see Figure 7. From here on we consider that particular choice when we refer to the sparse REFIRE network.

Remarkably, the real part of all eigenvalues is less than or equal to one, Figure 3 right, indicating stability of network dynamics. Although equation (7) guarantees that $n$ eigenvalues are equal to one, it does not rule out the existence of eigenvalues with greater real part. We speculate that the absence of such eigenvalues in the spectrum is due to the $l_1$ term in equation (9), the minimization of which could be viewed as a shrinkage of Gershgorin circles. We find that the connectivity learned

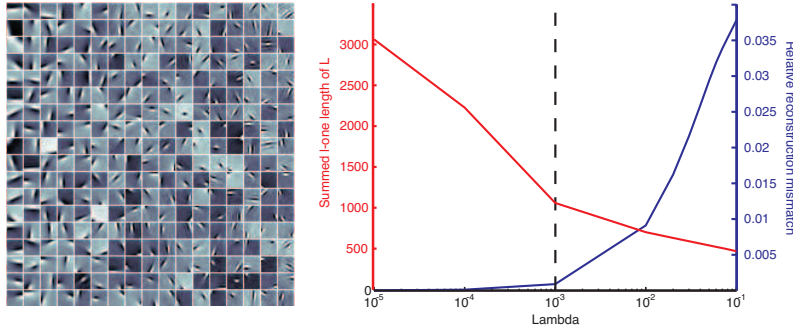

Figure 2: The sparse REFIRE network. Left: the patches corresponding to columns of $D$ sorted by variance. Right: Summed $l_1$-norm of all columns of L (left y-axis, red), the reconstruction mismatch $|(D - DL)|/|D|$ (right y-axis, blue) as a function of $\lambda$. Dashed line indicates the value of $\lambda$ chosen for the sparse REFIRE network.

was asymmetric with substantial imaginary components in the eigenvalues, see Figure 3 right. In general, the sparse REFIRE network is unlikely to be symmetric because the connection weights between a pair of neurons are not decided based solely on the identity of the neurons in the pair but are dependent on other connections of the same pre-synaptic neuron.

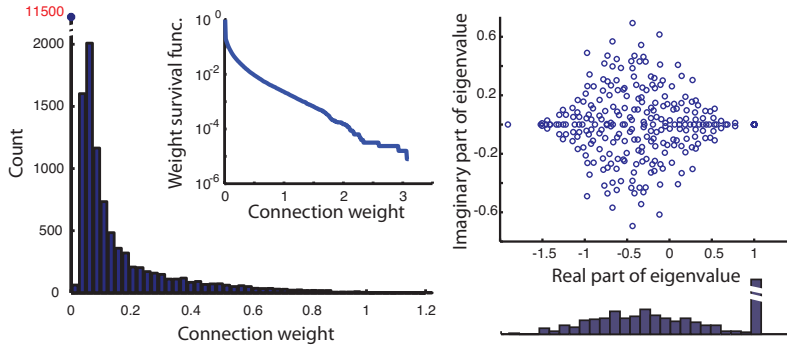

Figure 3: Properties of lateral connections. Left: distribution of lateral connectivity weights. Inset shows a survival plot with logarithmic y-axis and same axes limits. Right: scatter plot of eigenvalues of the lateral connectivity matrix. Note that there are many eigenvalues at real value one, imaginary value zero. Histogram shown below plot

Numerical simulations of the dynamics of a recurrent network with connectivity matrix $L$ confirm that the percept remains stable during the network dynamics. We chose an image patch at random and simulated the network dynamics. As can be seen in Figure 4 left, despite significant changes in the activity of the neurons, the percept encoded by the network remained stable, PSNR between original image and image after dynamics lasting 100 neuronal time constants: 45.5dB. The dynamics of the network desparsified the representation (Figure 4 right). Averaged across multiple patches, the value of each coefficient in the sparse representation was 0.0704, while after the network dynamics this increased to 0.0752, though still below the value obtained for the frame coefficients representation which was 0.0814.

## 3.2 Computational advantages of the sparse REFIRE network

In this section, we consider possible computational advantages for the de-coupling between the sensory percept and it representation by neuronal activity. Specifically, we address a shortcoming of the sparse representation, its lack of robustness [13]. Namely, the fact that stimuli that differ only to a small degree might end up being represented with very different coefficients. Intuitively speaking, this may occur when two (or more) dictionary elements compete for the same role in the

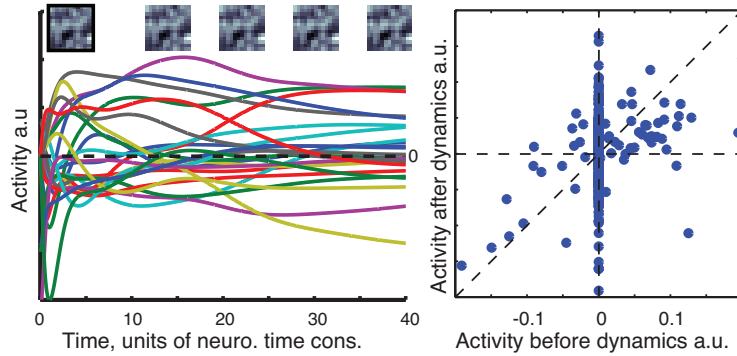

Figure 4: Evolution of neuronal activity in time. Left: activity of a subset of neurons over time. Top shows the original percept (framed in black) and plotted left to right patches taken from consecutive points in the dynamics. Right: scatter of the coefficients before and after 400 neuronal time constants of the dynamics.

sparse representation. To arrive at a sparse approximation of the stimuli either one of the dictionary elements could potentially be used, but due to the high cost of non-sparseness both of them together are not likely to be chosen in a given representation. Thus, small changes in the image, as might arise due to various noise sources, might cause one of the coefficients to be preferred over the other in an essentially random fashion, potentially resulting in very different coefficient values for highly similar images.

The dynamics of the sparse REFIRE network improve the robustness of the coefficient values in the face of noise. In order to model this effect we extract a single patch and corrupt it repeatedly with i.i.d 5% Gaussian noise. Figure 5 left shows two patches with similar orientation. Figure 5 middle shows the values of these two coefficients for the sparse approximation taken across the different noise repetitions. As can be clearly seen only one or the other of the two coefficients is used, exemplifying the competition described above. The resulting flickering in the coefficients exemplifies this lack of robustness. Note that the true lack of robustness arises due to multicollinear relations between the different dictionary elements. Here we restrict ourselves to two in the interests of clarity. Figure 5 right shows these coefficient values plotted one against the other in red along with the values of the two coefficients following the model dynamics in blue. In the latter case, the coefficient values between different repetitions remain fairly constant and the flickering representation as in Figure 5 middle is abolished.

We further examined the utility of a more stable representation by training a Naive Bayes classifier to discriminate between noisy versions of two patches. We corrupt the two patches with i.i.d noise and train the classifier on 75% of the data while reserving the remaining data for testing generalization. We train one of classifier on the sparse representation and the other on the representation following the dynamics of the sparse REFIRE network. We find that the generalization of the classifier learned following the dynamics was indeed higher, providing 92% accuracy, while the sparse coefficient trained classifier scored 83% accuracy.

We then demonstrate the computational advantages of the sparse REFIRE network in a more realistic scenario, encoding a set of patches extracted from an image by shifting the patch one pixel at a time. Such a shift can be caused by fixational drift or slow self-movement. Figure 5 right top shows a subset of the patches extracted in this fashion. For each of the patches we calculate the sparse approximation coefficients and then determine the dot product between the representation of consecutive patches. We then take the same coefficients, evolve them through the dynamics of the sparse REFIRE network network and compute the dot product between these new coefficients. Figure 5 right bottom shows the normalized dot product, the value of the dot product between the coefficients of two consecutive patches after the sparse REFIRE network dynamics, divided by the same dot product between the original coefficients. As can be seen, for nearly all cases the ratio is higher than one, indicating a smoother transition between the coefficients of the consecutive patches.

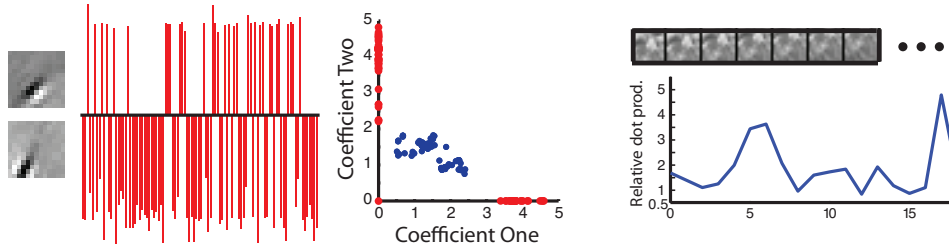

Figure 5: Sparse REFIRE network dynamics enhances the robustness of representation. Left: the patches corresponding to two columns of $D$ with similar tuning. Followed by the coefficient of each of the patch in the representation of the different noisy image instantiations and a scatter plot of the coefficient values before recurrent dynamics (red) and following (blue) recurrent dynamics. Right: an example of the patches in the sliding frame (top) and the normalized dot product between consecutive patches.

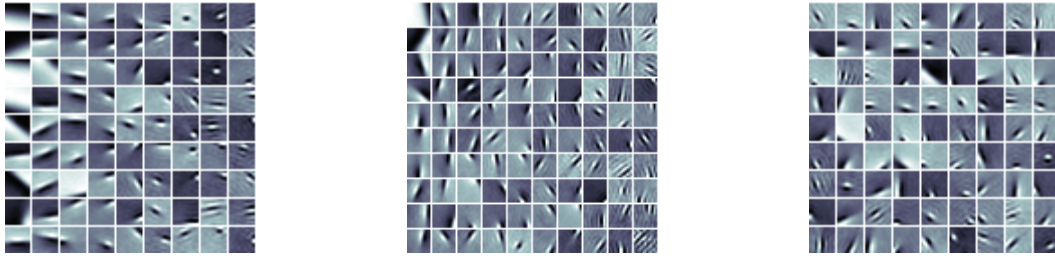

Figure 6: Dictionary clustering. Clusters of patches obtained by a three-way sparse REFIRE network partitioning by normalized cut. Note the mainly horizontal orientation of the first set of patches and the vertical orientation of the second.

The sparse REFIRE network encodes useful information regarding the relation between the different dictionary elements. This can be probed by partitioning performed on the graph [20]. Figure 6 shows the components of a normalized cut performed on the sparse REFIRE network. The left group shows clear bias towards horizontal orientation tuning, the middle towards vertical. Thus, subspaces can be learned directly from partitioning on the sparse REFIRE network offering a complementary approach to learning structured models directly from the data [26, 27].

Finally, the sparse REFIRE network serves as an integrator of the sensory input. Eigenspace of the unit eigenvalue is a multi-dimensional generalization of the line attractor used to model persistent activity [16]. However, unlike the persistent activity theory, which focuses on dynamics along the line attractor, we emphasize the transient dynamics approaching the unitary eigenspace.

## 4 Discussion

This study makes a number of novel contributions. First, we propose and demonstrate that in an over-complete representation certain types of network connectivity allow the percept, i.e. the stimulus represented by the network activity, to remain fixed in time despite changing neuronal activity. Second, we propose the sparse REFIRE network as a biologically plausible model for cortical lateral connections that enables such persistent percepts. Third, we point out that the ability to manipulate activity without affecting the accuracy of representation can be exploited in order to achieve computational goals. As an example, we show that the sparse REFIRE network dynamics, though causing the representation to be less sparse, alleviates the problem of representation non-robustness.

Although this study focused on sensory representation in the visual cortex, the framework can be extended to other sensory modalities, motor cortex and, perhaps, even higher cognitive areas such as prefrontal cortex or hippocampus.

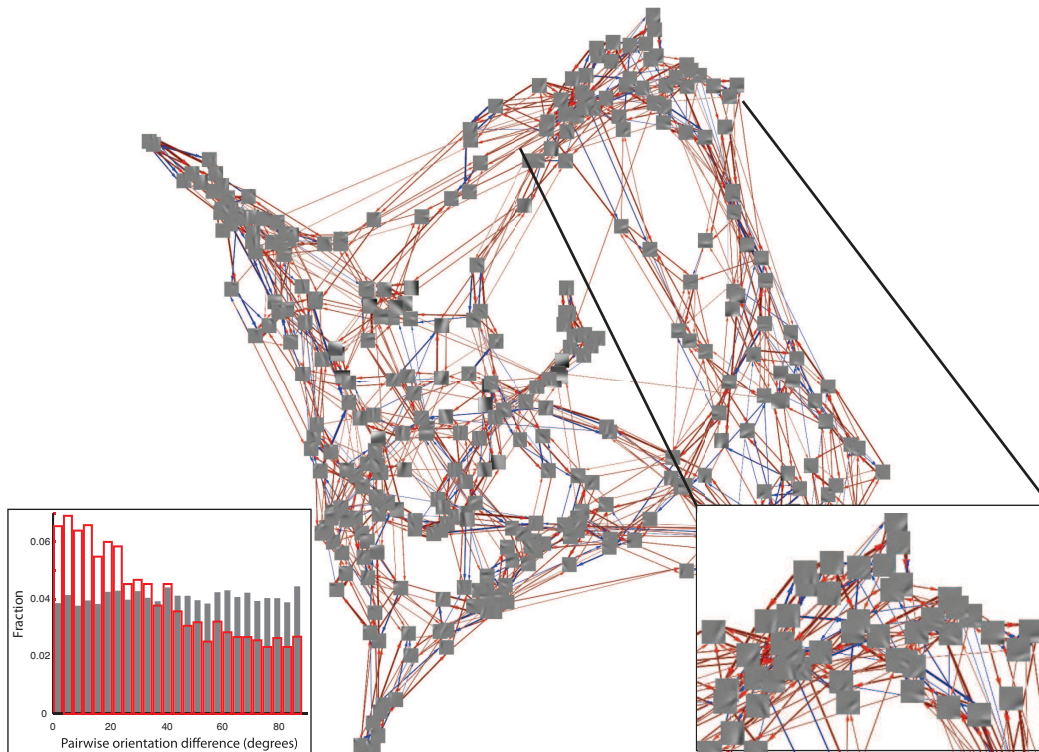

Figure 7: sparse REFIRE network structure. Nodes are shown by a patch corresponding to its feature vector. Arrows indicate connections, blue excitatory, red inhibitory. Plot organized to put strongly connected nodes close in space. Only strongest connections shown in the interests of clarity. Inset: Left: histogram of connectivity fraction by difference in feature orientation; red non-zero connections, gray all connections. Right: zoomed in view.

The sparse REFIRE network model bears an important relation to the family of sparse subspace models, which have been suggested to improve the robustness of sparse representations[26, 27]. We have shown that subspaces can be learned directly from the graph by standard graph partitioning algorithms. The optimal way to leverage the information embodied in the sparse REFIRE network to learn subspace-like models is a subject of ongoing work with promising results as is the study of different matrices $L$ that allow persistent percepts.

## Acknowledgments

We would like to thank Anatoli Grinshpan, Tao Hu, Alexei Koulakov, Bruno Olshausen and Lav Varshney for fruitful discussions and Frank Midgley for assistance with preparing figure 7.

## References

[1] B. A. Olshausen and D. J. Field, "Emergence of simple-cell receptive field properties by learning a sparse code for natural images," *Nature*, vol. 381, pp. 607–9, Jun 1996.

[2] M. Rehn and F. Sommer, "A network that uses few active neurones to code visual input predicts the diverse shapes of cortical receptive fields," *Journal of Computational Neuroscience*, vol. 22, pp. 135–146, 2007. 10.1007/s10827-006-0003-9.

[3] P. J. Garrigues and B. A. Olshausen, "Learning horizontal connections in a sparse coding model of natural images," *Advances in Neural Information Processing Systems*, vol. 20, pp. 505–512, 2008.

[4] O. Shriki, H. Sompolinsky, and D. D. Lee, "An information maximization approach to over-complete and recurrent representations," *Advances in Neural Information Processing Systems*, vol. 12, pp. 87–93, 2000.

[5] D. B. Chklovskii and A. A. Koulakov, "Maps in the brain: What can we learn from them?," *Annual Review of Neuroscience*, vol. 27, no. 1, pp. 369–392, 2004.

[6] G. Major and D. Tank, "Persistent neural activity: prevalence and mechanisms," *Current opinion in neurobiology*, vol. 14, no. 6, pp. 675–684, 2004.

[7] M. Goldman, "Memory without feedback in a neural network," *Neuron*, vol. 61, no. 4, pp. 621–634, 2009.

[8] A. Hyvarinen, J. Hurri, and P. O. Hoyer, *Natural Image Statistics: A Probabilistic Approach to Early Computational Vision.* Springer Publishing Company, Incorporated, 2009.

[9] O. Christensen, *An Introduction to Frames and Riesz Bases.* birkhauser, 2003.

[10] S. Mallat and Z. Zhang, "Matching pursuits with time-frequency dictionaries," *Signal Processing, IEEE Transactions on*, vol. 41, pp. 3397 –3415, dec 1993.

[11] S. Chen, D. Donoho, and M. Saunders, "Atomic decomposition by basis pursuit," *SIAM review*, vol. 43, no. 1, pp. 129–159, 2001.

[12] R. Tibshirani, "Regression shrinkage and selection via the lasso," *Journal of the Royal Statistical Society (Series B)*, vol. 58, pp. 267–288, 1996.

[13] C. J. Rozell, D. H. Johnson, R. G. Baraniuk, and B. A. Olshausen, "Sparse coding via thresholding and local competition in neural circuits," *Neural Comput*, vol. 20, pp. 2526–63, 2008.

[14] V. Braitenberg and A. Schüz, *Cortex: Statistics and Geometry of Neuronal Connectivity*. Berlin, Germany: Springer, 1998. ISBN: 3-540-63816-4.

[15] S. Cannon, D. Robinson, and S. Shamma, "A proposed neural network for the integrator of the oculomotor system," *Biological Cybernetics*, vol. 49, no. 2, pp. 127–136, 1983.

[16] H. Seung, "How the brain keeps the eyes still," *Proceedings of the National Academy of Sciences*, vol. 93, no. 23, p. 13339, 1996.

[17] J. Kovacevic and A. Chebira, "An introduction to frames," *Found. Trends Signal Process.*, vol. 2, no. 1, pp. 1–94, 2008.

[18] Y. Mishchenko, T. Hu, J. Spacek, J. Mendenhall, K. M. Harris, and D. B. Chklovskii, "Ultrastructural analysis of hippocampal neuropil from the connectomics perspective," *Neuron*, vol. 67, no. 6, pp. 1009–1020, 2010.

[19] L. R. Varshney, P. J. Sjöström, and D. B. Chklovskii, "Optimal information storage in noisy synapses under resource constraints," *Neuron*, vol. 52, no. 3, pp. 409 – 423, 2006.

[20] B. Cheng, J. Yang, S. Yan, Y. Fu, and T. Huang, "Learning with L1-Graph for Image Analysis," *IEEE Transactions on Image Processing*, p. 1, 2010.

[21] E. Elhamifar and R. Vidal, "Sparse subspace clustering," in *CVPR*, pp. 2790 –2797, 2009.

[22] R. Jenatton, J. Mairal, G. Obozinski, and F. Bach, "Proximal Methods for Sparse Hierarchical Dictionary Learning," *Proc. ICML*, 2010.

[23] D. Zoran and Y. Weiss, "The" Tree-Dependent Components" of Natural Images are Edge Filters," *Advances in Neural Information Processing Systems*, 2009.

[24] J. Mairal, F. Bach, J. Ponce, and G. Sapiro, "Online learning for matrix factorization and sparse coding," *Journal of Machine Learning Research*, vol. 11, pp. 19–60, 2010.

[25] S. Song, P. J. Sjöström, M. Reigl, S. Nelson, and D. B. Chklovskii, "Highly nonrandom features of synaptic connectivity in local cortical circuits," *PLoS Biol*, vol. 3, p. e68, Mar 2005.

[26] G. Yu, G. Sapiro, and S. Mallat, "Image modeling and enhancement via structured sparse model selection," 2010.

[27] K. Kavukcuoglu, M. Ranzato, R. Fergus, and Y. LeCun, "Learning invariant features through topographic filter maps," in *Proc. International Conference on Computer Vision and Pattern Recognition (CVPR'09)*, IEEE, 2009.

